# An MDP-Based Approach to Online Mechanism Design

**David C. Parkes**
Division of Engineering and Applied Sciences
Harvard University
parkes@eecs.harvard.edu

**Satinder Singh**
Computer Science and Engineering
University of Michigan
baveja@umich.edu

## Abstract

Online mechanism design (MD) considers the problem of providing incentives to implement desired system-wide outcomes in systems with self-interested agents that arrive and depart dynamically. Agents can choose to misrepresent their arrival and departure times, in addition to information about their value for different outcomes. We consider the problem of maximizing the total long-term value of the system despite the self-interest of agents. The online MD problem induces a Markov Decision Process (MDP), which when solved can be used to implement optimal policies in a truth-revealing Bayesian-Nash equilibrium.

## 1 Introduction

Mechanism design (MD) is a subfield of economics that seeks to implement particular outcomes in systems of rational agents [1]. Classically, MD considers static worlds in which a one-time decision is made and all agents are assumed to be patient enough to wait for the decision. By contrast, we consider dynamic worlds in which agents may arrive and depart over time and in which a sequence of decisions must be made without the benefit of hindsight about the values of agents yet to arrive. The MD problem for dynamic systems is termed *online mechanism design* [2]. Online MD supposes the existence of a center, that can receive messages from agents and enforce a particular outcome and collect payments.

Sequential decision tasks introduce new subtleties into the MD problem. First, decisions now have expected value instead of certain value because of uncertainty about the future. Second, new temporal strategies are available to an agent, such as waiting to report its presence to try to improve its utility within the mechanism. Online mechanisms must bring truthful and immediate revelation of an agent's value for sequences of decisions into equilibrium.

Without the problem of private information and incentives, the sequential decision problem in online MD could be formulated and solved as a Markov Decision Process (MDP). In fact, we show that an optimal policy and MDP-value function can be used to define an online mechanism in which truthful and immediate revelation of an agent's valuation for different sequences of decisions is a Bayes-Nash equilibrium.

Our approach is very general, applying to any MDP in which the goal is to maximize the total expected sequential value across all agents. To illustrate the flexibility of this model, we can consider the following illustrative applications:

**reusable goods.** A renewable resource is available in each time period. Agents arrive and submit a bid for a particular quantity of resource for each of a contiguous sequence of periods, and before some deadline.

**multi-unit auction.** A finite number of identical goods are for sale. Agents submit bids for a quantity of goods with a deadline, by which time a winner-determination decision must be made for that agent.

**multiagent coordination.** A central controller determines and enforces the actions that will be performed by a dynamically changing team of agents. Agents are only able to perform actions while present in the system.

Our main contribution is to identify this connection between online MD and MDPs, and to define a new family of dynamic mechanisms, that we term the *online VCG mechanism*. We also clearly identify the role of the ability to *stall* a decision, as it relates to the value of an agent, in providing for Bayes-Nash truthful mechanisms.

## 1.1  Related Work

The problem of online MD is due to Friedman and Parkes [2], who focused on strategyproof online mechanisms in which immediate and truthful revelation of an agent's valuation function is a *dominant strategy* equilibrium. The authors define the mechanism that we term the *delayed* VCG mechanism, identify problems for which the mechanism is strategyproof, and provide the seeds of our work in Bayes-Nash truthful mechanisms. Work on online auctions [3] is also related, in that it considers a system with dynamic agent arrivals and departures. However, the online auction work considers a much simpler setting (see also [4]), for instance the allocation of a fixed number of identical goods, and places less emphasis on temporal strategies or allocative efficiency. Awerbuch et al. [5], provide a general method to construct online auctions from online optimization algorithms. In contrast to our methods, their methods consider the special case of single-minded bidders with a value $v_i$ for a particular set of resources $r_i$, and are only temporally strategyproof in the special-case of online algorithms with a non-decreasing acceptance threshold.

## 2  Preliminaries

In this section, we introduce a general discrete-time and finite-action formulation for a multiagent sequential decision problem. Putting incentives to one side for now, we also define and solve an MDP formalization of the problem. We consider a finite-horizon problem[1] with a set $T$ of discrete time points and a sequence of decisions $k = \{k_1, \ldots, k_T\}$, where $k_t \in K_t$ and $K_t$ is the set of feasible decisions in period $t$. Agent $i \in \mathcal{I}$ arrives at time $a_i \in T$, departs at time $d_i \in T$, and has value $v_i(k) \geq 0$ for the sequence of decisions $k$. By assumption, an agent has no

value for decisions outside of interval $[a_i, d_i]$. Agents also face payments, which we allow in general to be collected after an agents departure. Collectively, information $\theta_i = (a_i, d_i, v_i)$ defines the type of agent $i$ with $\theta_i \in \Theta$. Agent types are sampled i.i.d. from a probability distribution $f(\theta)$, assumed known to the agents and to the central mechanism. We allow multiple agents to arrive and depart at the same time. Agent $i$, with type $\theta_i$, receives utility $u_i(k, p; \theta_i) = v_i(k; \theta_i) - p$, for decisions $k$ and payment $p$. Agents are modeled as expected-utility maximizers. We adopt as our goal that of maximizing the expected total sequential value across all agents.

If we were to simply ignore incentive issues, the expected-value maximizing decision problem *induces* an MDP. The state[2] of the MDP at time $t$ is the history-vector $h_t = (\theta_1, \ldots, \theta_t; k_1, \ldots, k_{t-1})$, and includes the reported types up to and including period $t$ and the decisions made up to and including period $t - 1$. The set of all possible states at time $t$ is denoted $H_t$. The set of all possible states across all time is $H = \bigcup_{t=1}^{T+1} H_t$. The set of decisions available in state $h_t$ is $K_t(h_t)$. Given a decision $k_t \in K_t(h_t)$ in state $h_t$, there is some probability distribution $Prob(h_{t+1}|h_t, k_t)$ over possible next states $h_{t+1}$ determined by the random new agent arrivals, agent departures, and the impact of decision $k_t$. This makes explicit the dynamics that were left implicit in type distribution $\theta_i \in f(\theta_i)$, and includes additional information about the domain.

The objective is to make decisions to maximize the expected total value across all agents. We define a payoff function for the induced MDP as follows: there is a payoff $R^i(h_t, k_t) = v_i(k_{\leq t}; \theta_i) - v_i(k_{\leq t-1}; \theta_i)$, that becomes available from agent $i$ upon taking action $k_t$ in state $h_t$. With this, we have $\sum_{t=1}^{\tau} R^i(h_t; k_t) = v_i(k_{\leq \tau}; \theta_i)$, for all periods $\tau$. The summed value, $\sum_i R^i(h_t, k_t)$, is the payoff obtained from all agents at time $t$, and is denoted $R(h_t, k_t)$. By assumption, the reward to an agent in this basic online MD problem depends only on decisions, and not on state. The transition probabilities and the reward function defined above, together with the feasible decision space, constitute the induced MDP $M_f$.

Given a policy $\pi = \{\pi_1, \pi_2, \ldots, \pi_T\}$ where $\pi_t : H_t \to K_t$, an MDP defines an MDP-value function $V^\pi$ as follows: $V^\pi(h_t)$ is the expected value of the summed payoff obtained from state $h_t$ onwards under policy $\pi$, i.e., $V^\pi(h_t) = E_\pi\{R(h_t, \pi(h_t)) + R(h_{t+1}, \pi(h_{t+1})) + \cdots + R(h_T, \pi(h_T))\}$. An optimal policy $\pi^*$ is one that maximizes the MDP-value of every state[3] in $H$. The optimal MDP-value function $V^*$ can be computed via the following *value iteration* algorithm: for $t = T - 1, T - 2, \ldots, 1$

$$\forall h \in H_t \ \ V^*(h) = \max_{k \in K_t(h)} [R(h, k) + \sum_{h' \in H_{t+1}} Prob(h'|h, k)V^*(h')]$$

where $V^*(h \in H_T) = \max_{k \in K_T(h)} R(h, k)$. This algorithm works backwards in time from the horizon and has time complexity polynomial in the size of the MDP and the time horizon $T$.

Given the optimal MDP-value function, the optimal policy is derived as follows: for $t < T$

$$\pi^*(h \in H_t) = \arg \max_{k \in K_t(h)} [R(h, k) + \sum_{h' \in H_{t+1}} Prob(h'|h, k)V^*(h')]$$

and $\pi^*(h \in H_T) = \arg \max_{k \in K_T(h)} R(h, k)$. Note that we have chosen not to subscript the optimal policy and MDP-value by time because it is implicit in the length of the state.

Let $R_{<t'}(h_t)$ denote the total payoff obtained prior to time $t'$ for a state $h_t$ with $t \geq t'$. The following property of MDPs is useful.

**Lemma 1 (MDP value-consistency)** *For any time $t < T$, and for any policy $\pi$, $E_{\{h_{t+1},...,h_T|h_t,\pi\}}\{R_{<t'}(h_{t'}) + V^\pi(h_{t'})\} = R_{<t}(h_t) + V^\pi(h_t)$, for all $t' \geq t$, where the expectation is taken with respect to a (correct) MDP model, $M_f$, given information up to and including period $t$ and policy $\pi$.*

We will need to allow for incorrect models, $M_f$, because agents may misreport their true types $\theta$ as untruthful types $\hat{\theta}$. Let $h_t(\hat{\theta}; \pi)$ denote the state at time $t$ produced by following policy $\pi$ on agents with reported types $\hat{\theta}$. Payoff, $R(h_t, k_t)$, will always denote the payoff with respect to the reported valuations of agents; in particular, $R_{<t'}(\hat{\theta}; \pi)$ denotes the total payoff prior to period $t'$ obtained by applying policy $\pi$ to reported types $\hat{\theta}$.

**Example.** (WiFi at Starbucks) [2] There is a finite set of WiFi (802.11b) channels to allocate to customers that arrive and leave a coffee house. A decision defines an allocation of a channel to a customer for some period of time. There is a known distribution on agent valuations and a known arrival and departure process. Each customer has her own value function, for example "I value any 10 minute connection in the next 30 minutes a \$0.50." The decision space might include the ability to delay making a decision for a new customer, before finally making a definite allocation decision. At this point the MDP reward would be the *total* value to the agent for this allocation into the future.

The following *domain properties* are required to formally state the economic properties of our online VCG mechanism. First, we need *value-monotonicity*, which will be sufficient to provide for voluntary participation in our mechanism. Let $\theta_i \in h_t$ denote that agent $i$ with type $\theta_i$ arrived in some period $t' \leq t$ in history $h_t$.

**Definition 1 (value-monotonicity)** *MDP, $M_f$, satisfies value-monotonicity if for all states, $h_t$, the optimal MDP-value function satisfies $V^*(h_t(\hat{\theta} \cup \theta_i; \pi^*)) - V^*(h_t(\hat{\theta}; \pi^*)) \geq 0$, for agent $i$ with type $\theta_i$ that arrives in period $t$.*

Value-monotonicity requires that the arrival of each additional agent has a positive effect on the expected total value from that state forward. In WiFi at Starbucks, this is satisfied because an agent with a low value can simply be ignored by the mechanism. It may fail in other problems, for instance in a physical domain with a new robot that arrives and blocks the progress of other robots.

Second, we need *no-positive-externalities*, which will be sufficient for our mechanisms to run without payment deficits to the center.

**Definition 2 (no-positive-externalities)** *MDP, $M_f$, satisfies no-positive-externalities if for all states, $h_t$, the optimal MDP-value function satisfies $V^*(h_t(\hat{\theta} \cup \theta_i; \pi^*)) - v_i(\pi^*(h_t(\hat{\theta} \cup \theta_i; \pi^*)); \theta_i) \leq V^*(h_t(\hat{\theta}; \pi^*))$, for agent $i$ with type $\theta_i$ that arrives in period $t$.*

No-positive-externalities requires that the arrival of each additional agent can only make the other agents worse off in expectation. This holds in WiFi at Starbucks, because a new agent can take resources from other agents, but not in general, for instance when agents are both providers and consumers of resources or when multiple agents are needed to make progress.

# 3   The Delayed VCG Mechanism

In this section, we define the delayed VCG mechanism, which was introduced in Friedman and Parkes [2]. The mechanism implements a sequence of decisions based on agent reports but delays final payments until the final period $T$. We prove that the delayed VCG mechanism brings truth-revelation into a Bayes-Nash equilibrium in combination with an optimal MDP policy.

The delayed VCG mechanism is a direct-revelation online mechanism (DRM). The strategy space restricts an agent to making a single claim about its type. Formally, an online direct-revelation mechanism, $\mathcal{M} = (\Theta; \pi, p)$, defines a feasible type space $\Theta$, along with a decision policy $\pi = (\pi_1, \ldots, \pi_T)$, with $\pi_t : H_t \to K_t$, and a payment rule $p = (p_1, \ldots, p_T)$, with $p_t : H_t \to \mathbb{R}^N$, such that $p_{t,i}(h_t)$ denotes the payment to agent $i$ in period $t$ given state $h_t$.

**Definition 3 (delayed VCG mechanism)** *Given history* $h \in H$, *mechanism* $\mathcal{M}_{\text{Dvcg}} = (\Theta; \pi, p^{\text{Dvcg}})$, *implements decisions* $k_t = \pi(h_t)$, *and computes payment*

$$p_i^{\text{Dvcg}}(\hat{\theta}; \pi) = R_{\leq T}^i(\hat{\theta}; \pi) - \left[ R_{\leq T}(\hat{\theta}; \pi) - R_{\leq T}(\hat{\theta}_{-i}; \pi) \right] \tag{1}$$

*to agent* $i$ *at the end of the final period, where* $R_{\leq T}(\hat{\theta}_{-i}; \pi)$ *denotes the total reported payoff for the optimal policy in the system without agent* $i$.

An agent's payment is discounted from its reported value for the outcome by a term equal to the total (reported) marginal value generated by its presence. Consider agent $i$, with type $\theta_i$, and let $\theta_{<i}$ denote the types of agents that arrive before agent $i$, and let $\theta_{>i}$ denote a random variable (distributed according to $f(\theta)$) for the agents that arrive after agent $i$.

**Definition 4 (Bayesian-Nash Incentive-Compatible)** *Mechanism* $M_{\text{Dvcg}}$ *is Bayesian-Nash incentive-compatible if and only if the policy* $\pi$ *and payments satisfy:*

$$E_{\theta_{>i}}\{v_i(\pi(\theta_{<i}, \theta_i, \theta_{>i}); \theta_i) - p_i^{\text{Dvcg}}(\theta_{<i}, \theta_i, \theta_{>i}; \pi)\} \tag{BNIC}$$
$$\geq E_{\theta_{>i}}\{v_i(\pi(\theta_{<i}, \hat{\theta}_i, \theta_{>i}); \theta_i) - p_i^{\text{Dvcg}}(\theta_{<i}, \hat{\theta}_i, \theta_{>i}; \pi)\}$$

*for all types* $\theta_{<i}$, *all types* $\theta_i$, *and all* $\hat{\theta}_i \neq \theta_i$.

Bayes-Nash IC states that truth-revelation is utility maximizing in expectation, given common knowledge about the distribution on agent valuations and arrivals $f(\theta)$ and when other agents are truthful. Moreover, it implies immediate revelation, because the type includes information about an agent's arrival period.

**Theorem 1** *A delayed VCG mechanism,* $(\Theta; \pi^*, p^{\text{Dvcg}})$, *based on an optimal policy* $\pi^*$ *for a correct MDP model defined for a decision space that includes stalling is Bayes-Nash incentive compatible.*

**Proof.**   Assume without loss of generality that the other agents are reporting truthfully. Consider some agent $i$, with type $\theta_i$, and suppose agents $\theta_{<i}$ have already arrived. Now, the expected utility to agent $i$ when it reports type $\hat{\theta}_i$, substituting for the payment term $p_i^{\text{Dvcg}}$, is $E_{\theta_{>i}}\{v_i(\pi^*(\theta_{<i}, \hat{\theta}_i, \theta_{>i}); \theta_i) + \sum_{j \neq i} R_{\leq T}^j(\theta_{<i}, \hat{\theta}_i, \theta_{>i}; \pi^*) - R_{\leq T}(\theta_{<i}, \theta_{>i}; \pi^*)\}$. We can ignore the final term because it does not depend on the choice of $\hat{\theta}_i$ at all. Let $\tau$ denote the arrival period $a_i$ of agent $i$, with state $h_\tau$ including agent types $\theta_{<i}$, decisions up to and including period $\tau - 1$, and the reported type of agent $i$ if it makes a report in period

$a_i$. Ignoring $R_{<\tau}(h_\tau)$, which is the total payoff already received by agents $j \neq i$ in periods up to and including $\tau - 1$, the remaining terms are equal to the expected value of the summed payoff obtained from state $h_\tau$ onwards under policy $\pi^*$, $E_{\pi^*}\{v_i(\pi^*(h_\tau); \theta_i) + \sum_{j \neq i} v_j(\pi^*(h_\tau); \hat{\theta}_j) + v_i(\pi^*(h_{\tau+1}); \theta_i) + \sum_{j \neq i} v_j(\pi^*(h_{\tau+1}); \hat{\theta}_j) + \ldots + v_i(\pi^*(h_T); \theta_i) + \sum_{j \neq i} v_j(\pi^*(h_T); \hat{\theta}_j)\}$, defined with respect to the true type of agent $i$ and the reported types of agents $j \neq i$. This is the MDP-value for policy $\pi^*$ in state $h_\tau$, $E_{\pi^*}\{R(h_\tau, \pi^*(h_\tau)) + R(h_{\tau+1}, \pi^*(h_{\tau+1})) + \ldots + R(h_T, \pi^*(h_T))\}$, because agents $j \neq i$ are assumed to report their true types in equilibrium. We have a contradiction with the optimality of policy $\pi^*$ because if there is some type $\hat{\theta}_i \neq \theta_i$ that agent $i$ can report to improve the MDP-value of policy $\pi^*$, given types $\theta_{<i}$, then we can construct a new policy $\pi'$ that is better than policy $\pi^*$; policy $\pi'$ is identical to $\pi^*$ in all states except $h_\tau$, when it implements the decision defined by $\pi^*$ in the state with type $\theta_i$ replaced by type $\hat{\theta}_i$. The new policy, $\pi'$, lies in the space of feasible policies because the decision space includes stalling and can mimic the effect of any manipulation in which agent $i$ reports a later arrival time. $\qquad\square$

The effect of the first term in the discount in Equation 1 is to align the agent's incentives with the system-wide objective of maximizing the total value across agents. We do not have a stronger equilibrium concept than Bayes-Nash because the mechanism's model will be incorrect if other agents are not truthful and its policy suboptimal. This leaves space for useful manipulation. The following corollary captures the requirement that the MDPs decision space must allow for stalling, i.e. it must include the option to delay making a decision that will determine the value of agent $i$ until some period after the agent's arrival. Say an agent has *patience* if $d_i > a_i$.

**Corollary 2** *A delayed VCG mechanism cannot be Bayes-Nash incentive-compatible if agents have any patience and the expected value of its policy can be improved by stalling a decision.*

If the policy can be improved through stalling, then an agent can improve its expected utility by delaying its reported arrival to correct for this, and make the policy stall. This delayed VCG mechanism is *ex ante* **efficient**, because it implements the policy that maximizes the expected total sequential value across all agents. Second, it is *interim* **individual-rational** as long as the MDP satisfies the *value-monotonicity* property. The expected utility to agent $i$ in equilibrium is $E_{\theta > i}\{R_{\leq T}(\theta_{<i}, \theta_i, \theta_{>i}; \pi^*) - R_{\leq T}(\theta_{<i}, \theta_{>i}; \pi^*)\}$, which is equivalent to value-monotonicity. Third, the mechanism is *ex ante* **budget-balanced** as long as the MDP satisfies the *no-positive-externalities* property. The expected payment by agent $i$, with type $\theta_i$, to the mechanism is $E_{\theta > i}\{R_{\leq T}(\theta_{<i}, \theta_{>i}; \pi^*) - (R_{\leq T}(\theta_{<i}, \theta_i, \theta_{>i}; \pi^*) - R^i_{\leq T}(\theta_{<i}, \theta_i, \theta_{>i}; \pi^*))\}$, which is non-negative exactly when the no-positive-externalities condition holds.

## 4    The Online VCG Mechanism

We now introduce the *online* VCG mechanism, in which payments are determined as soon as all decisions are made that affect an agent's value. Not only is this a better fit with the practical needs of online mechanisms, but the online VCG mechanism also enables better computational properties than the delayed mechanism.

Let $V^\pi(h_t(\hat{\theta}_{-i}; \pi))$ denote the MDP-value of policy $\pi$ in the system without agent $i$, given reports $\theta_{-i}$ from other agents, and evaluated in some period $t$.

**Definition 5 (online VCG mechanism)** *Given history* $h \in H$, *mechanism* $\mathcal{M}_{\text{vcg}} = (\Theta; \pi, p_{\text{vcg}})$ *implements decisions* $k_t = \pi(h_t)$, *and computes payment*

$$p_i^{\text{vcg}}(\hat{\theta}; \pi) = R_{\leq m_i}^i(\hat{\theta}; \pi) - \left[ V^\pi(h_{\hat{a}_i}(\hat{\theta}; \pi)) - V^\pi(h_{\hat{a}_i}(\hat{\theta}_{-i}; \pi)) \right] \qquad (2)$$

*to agent $i$ in its commitment period $m_i$, with zero payments in all other periods.*

Note the payment is computed in the commitment period for an agent, which is some period before an agent's departure at which its value is fully determined. In WiFi at Starbucks, this can be the period in which the mechanism commits to a particular allocation for an agent.

Agent $i$'s payment in the online VCG mechanism is equal to its reported value from the sequence of decisions made by the policy, discounted by the expected marginal value that agent $i$ will contribute to the system (as determined by the MDP-value function for the policy in its arrival period). The discount is defined as the expected forward looking effect the agent will have on the value of the system. Establishing incentive-compatibility requires some care because the payment now depends on the stated arrival time of an agent. We must show that there is no systematic dependence that an agent can use to its advantage.

**Theorem 3** *An online VCG mechanism, $(\Theta; \pi^*, p^{\text{vcg}})$, based on an optimal policy $\pi^*$ for a correct MDP model defined for a decision space that includes stalling is Bayes-Nash incentive compatible.*

**Proof.** We establish this result by demonstrating that the expected value of the payment by agent $i$ in the online VCG mechanism is the same as in the delayed VCG mechanism, when other agents report their true types and for *any* reported type of agent $i$. This proves incentive-compatibility, because the policy in this online VCG mechanism is exactly that in the delayed VCG mechanism (and so an agent's value from decisions is the same), and with identical expected payments the equilibrium follows from the truthful equilibrium of the delayed mechanism. The first term in the payment (see Equation 2) is $R_{\leq m_i}^i(\hat{\theta}_i, \theta_{-i}; \pi^*)$ and has the same value as the first term, $R_{\leq T}^i(\hat{\theta}_i, \theta_{-i}; \pi^*)$, in the payment in the delayed mechanism (see Equation 1). Now, consider the discount term in Equation 2, and rewrite this as:

$$V^*(h_{\hat{a}_i}(\hat{\theta}_i, \theta_{-i}; \pi^*)) + R_{\hat{a}_i}(\theta_{-i}; \pi^*) - V^*(h_{\hat{a}_i}(\theta_{-i}; \pi^*)) - R_{\hat{a}_i}(\theta_{-i}; \pi^*) \qquad (3)$$

The expected value of the left-hand pair of terms in Equation 3 is equal to $V^*(h_{\hat{a}_i}(\hat{\theta}_i, \theta_{-i}; \pi^*)) + R_{\hat{a}_i}(\hat{\theta}_i, \theta_{-i}; \pi^*)$ because agent $i$'s announced type has no effect on the reward before its arrival. Applying Lemma 1, the expected value of these terms is *constant* and equal to the expected value of $V^*(h_{t'}(\hat{\theta}_i, \theta_{-i}; \pi^*)) + R_{t'}(\hat{\theta}_i, \theta_{-i}; \pi^*)$ for all $t' \geq a_i$ (with the expectation taken wrt history $h_{a_i}$ available to agent $i$ in its true arrival period.) Moreover, taking $t'$ to be the final period, $T$, this is also equal to the expected value of $R_{\leq T}(\hat{\theta}_i, \theta_{-i}; \pi^*)$, which is the expected value of the first term of the discount in the payment in the delayed VCG mechanism. Similarly, the (negated) expected value of the right-hand pair of terms in Equation 3 is constant, and equals $V^*(h_{t'}(\theta_{-i}; \pi^*)) + R_{t'}(\theta_{-i}; \pi^*)$ for all $t' \geq a_i$. Again, taking $t'$ to be the final period $T$ this is also equal to the expected value of $R_{\leq T}(\theta_{-i}; \pi^*)$, which is the expected value of the second term of the discount in the payment in the delayed VCG mechanism. $\square$

We have demonstrated that although an agent can systematically reduce the expected value of each of the first and second terms in the discount in its payment (Equation 2) by delaying its arrival, these effects exactly cancel each other out.

Note that it also remains important for incentive-compatibility on the online VCG mechanism that the policy allows stalling.

The online VCG mechanism shares the properties of allocative **efficiency** and **budget-balance** with the delayed VCG mechanism (under the same conditions). The online VCG mechanism is *ex post* **individual-rational** so that an agent's expected utility is always non-negative, a slightly stronger condition that for the delayed VCG mechanism. The expected utility to agent $i$ is $V^*(h_{a_i}) - V^*(h_{a_i} \setminus i)$ and non-negative because of the value-monotonicity property of MDPs.

The online VCG mechanism also suggests the possibility of new computational speed-ups. The payment to an agent only requires computing the optimal-MDP value without the agent in the state in which it arrives, while the delayed VCG payment requires computing the sequence of decisions that the optimal policy would have made in the counterfactual world without the presence of each agent.

## 5  Discussion

We described a direct-revelation mechanism for a general sequential decision making setting with uncertainty. In the Bayes-Nash equilibrium each agent truthfully reveals its private type information, and immediately upon arrival. The mechanism induces an MDP, and implements the sequence of decisions that maximize the expected total value across all agents. There are two important directions in which to take this preliminary work. First, we must deal with the fact that for most real applications the MDP that will need to be solved to compute the decision and payment policies will be too big to be solved exactly. We will explore methods for solving large-scale MDPs approximately, and consider the consequences for incentive-compatibility. Second, we must deal with the fact that the mechanism will often have at best an incomplete and inaccurate knowledge of the distributions on agent-types. We will explore the interaction between models of learning and incentives, and consider the problem of adaptive online mechanisms.

**Acknowledgments**

This work is supported in part by NSF grant IIS-0238147.

## Footnotes

[1]The model can be trivially extended to consider infinite horizons if all agents share the same discount factor, but will require some care for more general settings.

[2]Using histories as state in the induced MDP will make the state space very large. Often, there will be some function $g$ for which $g(h)$ is a sufficient statistic for all possible states $h$. We ignore this possibility here.

[3]It is known that a deterministic optimal policy always exists in MDPs[6].

## References

[1] Matthew O. Jackson. Mechanism theory. In *The Encyclopedia of Life Support Systems*. EOLSS Publishers, 2000.

[2] Eric Friedman and David C. Parkes. Pricing WiFi at Starbucks– Issues in online mechanism design. Short paper, In *Fourth ACM Conf. on Electronic Commerce (EC'03)*, 240–241, 2003.

[3] Ron Lavi and Noam Nisan. Competitive analysis of incentive compatible on-line auctions. In *Proc. 2nd ACM Conf. on Electronic Commerce (EC-00)*, 2000.

[4] Avrim Blum, Vijar Kumar, Atri Rudra, and Felix Wu. Online learning in online auctions. In *Proceedings of the 14th Annual ACM-SIAM symposium on Discrete algorithms*, 2003.

[5] Baruch Awerbuch, Yossi Azar, and Adam Meyerson. Reducing truth-telling online mechanisms to online optimization. In *Proc. ACM Symposium on Theory of Computing (STOC'03)*, 2003.

[6] M. L. Puterman. *Markov decision processes : discrete stochastic dynamic programming*. John Wiley & Sons, New York, 1994.
